# Extended Grassmann Kernels for Subspace-Based Learning

**Jihun Hamm**
GRASP Laboratory
University of Pennsylvania
Philadelphia, PA 19104
jhham@seas.upenn.edu

**Daniel D. Lee**
GRASP Laboratory
University of Pennsylvania
Philadelphia, PA 19104
ddlee@seas.upenn.edu

## Abstract

Subspace-based learning problems involve data whose elements are linear subspaces of a vector space. To handle such data structures, Grassmann kernels have been proposed and used previously. In this paper, we analyze the relationship between Grassmann kernels and probabilistic similarity measures. Firstly, we show that the KL distance in the limit yields the Projection kernel on the Grassmann manifold, whereas the Bhattacharyya kernel becomes trivial in the limit and is suboptimal for subspace-based problems. Secondly, based on our analysis of the KL distance, we propose extensions of the Projection kernel which can be extended to the set of affine as well as scaled subspaces. We demonstrate the advantages of these extended kernels for classification and recognition tasks with Support Vector Machines and Kernel Discriminant Analysis using synthetic and real image databases.

## 1   Introduction

In machine learning problems the data often live in a vector space, typically a Euclidean space. However, there are many other kinds of non-Euclidean spaces suitable for data outside this conventional context. In this paper we focus on the domain where each data sample is a linear subspace of vectors, rather than a single vector, of a Euclidean space. Low-dimensional subspace structures are commonly encountered in computer vision problems. For example, the variation of images due to the change of pose, illumination, etc, is well-aproximated by the subspace spanned by a few "eigenfaces". More recent examples include the dynamical system models of video sequences from human actions or time-varying textures, represented by the linear span of the observability matrices [1, 14, 13].

Subspace-based learning is an approach to handle the data as a collection of subspaces instead of the usual vectors. The appropriate data space for the subspace-based learning is the Grassmann manifold $\mathcal{G}(m, D)$, which is defined as the set of $m$-dimensional linear subspaces in $\mathbb{R}^D$. In particular, we can define positive definite kernels on the Grassmann manifold, which allows us to treat the space as if it were a Euclidean space. Previously, the Binet-Cauchy kernel [17, 15] and the Projection kernel [16, 6] have been proposed and demonstrated the potential for subspace-based learning problems.

On the other hand, the subspace-based learning problem can be approached purely probabilistically. Suppose the set of vectors are i.i.d samples from an arbitrary probability distribution. Then it is possible to compare two such distributions of vectors with probabilistic similarity measures, such as the KL distance[1], the Chernoff distance, or the Bhattacharyya/Hellinger distance, to name a few [11, 7, 8, 18]. Furthermore, the Bhattacharyya affinity is indeed a positive definite kernel function on the space of distributions and have nice closed-form expressions for the exponential family [7].

In this paper, we investigate the relationship between the Grassmann kernels and the probabilistic distances. The link is provided by the probabilistic generalization of subspaces with a Factor Analyzer which is a Gaussian 'blob' that has nonzero volume along all dimensions.

Firstly, we show that the KL distance yields the Projection kernel on the Grassmann manifold in the limit of zero noise, whereas the Bhattacharyya kernel becomes trivial in the limit and is suboptimal for subspace-based problems. Secondly, based on our analysis of the KL distance, we propose an extension of the Projection kernel which is originally confined to the set of linear subspaces, to the set of affine as well as scaled subspaces.

We will demonstrate the extended kernels with the Support Vector Machines and the Kernel Discriminant Analysis using synthetic and real image databases. The proposed kernels show the better performance compared to the previously used kernels such as Binet-Cauchy and the Bhattacharyya kernels.

## 2 Probabilistic subspace distances and kernels

In this section we will consider the two well-known probabilistic distances, the KL distance and the Bhattacharyya distance, and establish their relationships to the Grassmann kernels. Although these probabilistic distances are not restricted to specific distributions, we will model the data distribution as the Mixture of Factor Analyzers (MFA) [4]. If we have $i = 1, ..., N$ sets in the data, then each set is considered as i.i.d. samples from the $i$-th Factor Analyzer

$$x \sim p_i(x) = \mathcal{N}(u_i, C_i), \quad C_i = Y_i Y_i' + \sigma^2 I_D, \tag{1}$$

where $u_i \in \mathbb{R}^D$ is the mean, $Y_i$ is a full-rank $D \times m$ matrix $(D > m)$, and $\sigma$ is the ambient noise level. The FA model is a practical substitute for a Gaussian distribution in case the dimensionality $D$ of the images is greater than the number of samples $n$ in a set. Otherwise it is impossible to estimate the full covariance $C$ nor invert it.

More importantly, we use the FA distribution to provide the link between the Grassmann manifold and the space of probabilistic distributions. In fact a linear subspace can be considered as the 'flattened' $(\sigma \to 0)$ limit of a zero-mean $(u_i = 0)$, homogeneous $(Y_i' Y_i = I_m)$ FA distribution. We will look at the limits of the KL distance and the Bhattacharyya kernel under this condition.

### 2.1 KL distance in the limit

The (symmetrized) KL distance is defined as

$$J_{\text{KL}}(p_1, p_2) = \int [p_1(x) - p_2(x)] \log \frac{p_1(x)}{p_2(x)} \, dx. \tag{2}$$

Let $C_i = \sigma^2 I + Y_i Y_i'$ be the covariance matrix, and define $\widetilde{Y}_i = Y_i(\sigma^2 I + Y_i' Y_i)^{-1/2}$, and $\widetilde{Z} = 2^{-1/2}[\widetilde{Y}_1 \ \widetilde{Y}_2]$. In this case the KL distance is

$$
\begin{aligned}
J_{\text{KL}}(p_1, p_2) &= \frac{1}{2}\text{tr}(-\widetilde{Y}_1'\widetilde{Y}_1 - \widetilde{Y}_2'\widetilde{Y}_2) + \frac{\sigma^{-2}}{2}\text{tr}(Y_1'Y_1 + Y_2'Y_2 - \widetilde{Y}_1'Y_2Y_2'\widetilde{Y}_1 - \widetilde{Y}_2'Y_1Y_1'\widetilde{Y}_2) \\
&+ \frac{\sigma^{-2}}{2}(u_1 - u_2)'\left(2I_D - \widetilde{Y}_1\widetilde{Y}_1' - \widetilde{Y}_2\widetilde{Y}_2'\right)(u_1 - u_2).
\end{aligned} \tag{3}
$$

Under the subspace condition $(\sigma \to 0, u_i = 0, Y_i'Y_i = I_m, \ i = 1, ..., N)$, the KL distance simplifies to

$$
\begin{aligned}
J_{\text{KL}}(p_1, p_2) &= \frac{1}{2}(-2\frac{m}{\sigma^2 + 1}) + \frac{\sigma^{-2}}{2}\left(2m - 2\frac{1}{\sigma^2 + 1}\text{tr}(Y_1'Y_2Y_2'Y_1)\right) \\
&= \frac{1}{2\sigma^2(\sigma^2 + 1)}(2m - 2\text{tr}(Y_1'Y_2Y_2'Y_1))
\end{aligned}
$$

We can ignore the multiplying factors which do not depend on $Y_1$ or $Y_2$, and rewrite the distance as

$$J_{\text{KL}}(p_1, p_2) = 2m - 2\text{tr}(Y_1'Y_2Y_2'Y_1).$$

We immediately realize that the distance $J_{\text{KL}}(p_1, p_2)$ coincides with the definition of the squared Projection distance [2, 16, 6], with the corresponding Projection kernel

$$k_{\text{Proj}}(Y_1, Y_2) = \text{tr}(Y_1'Y_2Y_2'Y_1). \tag{4}$$

## 2.2 Bhattacharyya kernel in the limit

Jebara and Kondor [7, 8] proposed the Probability Product kernel

$$k_{\text{Prob}}(p_1, p_2) = \int [p_1(x)\, p_2(x)]^\alpha \, dx \; (\alpha > 0) \tag{5}$$

which includes the Bhattacharyya kernel as a special case.

Under the subspace condition ($\sigma \to 0$, $u_i = 0$, $Y_i'Y_i = I_m$, $i = 1, ..., N$) the kernel $k_{\text{Prob}}$ becomes

$$k_{\text{Prob}}(p_1, p_2) = \pi^{(1-2\alpha)D} 2^{-\alpha D} \alpha^{-D/2} \frac{\sigma^{2\alpha(m-D)+D}}{(\sigma^2+1)^{\alpha m}} \det(I_{2m} - \widetilde{Y}'\widetilde{Y})^{-1/2} \tag{6}$$

$$\propto \det\left(I_m - \frac{1}{(2\sigma^2+1)^2} Y_1'Y_2Y_2'Y_1\right)^{-1/2}. \tag{7}$$

Suppose the two subspaces $\text{span}(Y_1)$ and $\text{span}(Y_2)$ intersect only at the origin, that is, the singular values of $Y_1'Y_2$ are strictly less than 1. In this case $k_{\text{Prob}}$ has a finite value as $\sigma \to 0$ and the inversion of (7) is well-defined. In contrast, the diagonal terms of $k_{\text{Prob}}$ become

$$k_{\text{Prob}}(Y_1, Y_1) = \det\left((1 - \frac{1}{(2\sigma^2+1)^2})I_m\right)^{-1/2} = \left(\frac{(2\sigma^2+1)^2}{4\sigma^2(\sigma^2+1)}\right)^{m/2}, \tag{8}$$

which diverges to infinity as $\sigma \to 0$. This implies that after normalizing the kernel by the diagonal terms, the resulting kernel becomes a trivial kernel

$$\tilde{k}_{\text{Prob}}(Y_i, Y_j) = \begin{cases} 1, & \text{span}(Y_i) = \text{span}(Y_j) \\ 0, & \text{otherwise} \end{cases} , \quad \text{as } \sigma \to 0. \tag{9}$$

The derivations are detailed in the thesis [5]. As we claimed earlier, the Probability Product kernel, including the Bhattacharyya kernel, loses its discriminating power as the distributions become close to subspaces.

## 3 Extended Projection Kernel

Based on the analysis of the previous section, we will extend the Projection kernel (4) to more general spaces than the Grassmann manifold in this section. We will examine the two directions of extension: from linear to affine, and from homogeneous to scaled subspaces.

### 3.1 Extension to affine subspaces

An affine subspace in $\mathbb{R}^D$ is a linear subspace with an 'offset' . In that sense a linear subspace is simply an affine subspace with a zero offset. Analogously to the (linear) Grassmann manifold, we can define an affine Grassmann manifold as the set of all $m$-dim affine subspaces in $\mathbb{R}^D$ space [2]. The affine span is defined from the orthonormal basis $Y \in \mathbb{R}^{D\times m}$ and an offset $u \in \mathbb{R}^D$ by

$$\text{aspan}(Y, u) \triangleq \{x \mid x = Yv + u, \; \forall v \in \mathbb{R}^m\}. \tag{10}$$

By definition, the representation of an affine space by $(Y, u)$ is not unique and there is an invariant condition for the equivalent of representations:

**Definition 1** (invariance to representations)**.**

$\text{aspan}(Y_1, u_1) = \text{aspan}(Y_2, u_2)$ *if and only if* $\text{span}(Y_1) = \text{span}(Y_2)$ *and* $Y_1^\perp(Y_1^\perp)'u_1 = Y_2^\perp(Y_2^\perp)'u_2$,

*where* $Y^\perp$ *is any orthonormal basis for the orthogonal complement of* $\text{span}(Y)$.

Similarly to the definition of Grassmann kernels [6], we can now formally define the affine Grassmann kernel as follows. Let $k : (\mathbb{R}^{m\times D} \times \mathbb{R}^D) \times (\mathbb{R}^{m\times D} \times \mathbb{R}^D) \to \mathbb{R}$ be a real valued symmetric function $k(Y_1, u_1, Y_2, u_2) = k(Y_2, u_2, Y_1, u_1)$.

**Definition 2.** *A real valued symmetric function $k$ is an affine Grassmann kernel if it is positive definite and invariant to different representations:*

$k(Y_1, u_1, Y_2, u_2) = k(Y_3, u_3, Y_4, u_4)$ *for any* $Y_1, Y_2, Y_3, Y_4$, *and* $u_1, u_2, u_3, u_4$ *such that* $\mathrm{aspan}(Y_1, u_1) = \mathrm{aspan}(Y_3, u_3)$ *and* $\mathrm{aspan}(Y_2, u_2) = \mathrm{aspan}(Y_4, u_4)$.

With this definition we check if the KL distance in the limit suggests an affine Grassmann kernel.

The KL distance with the homogeneity condition only $Y_1'Y_1 = Y_2'Y_2 = I_m$ becomes,

$$J_{\mathrm{KL}}(p_1, p_2) \quad \rightarrow \quad \frac{1}{2\sigma^2}\left[2m - 2\mathrm{tr}(Y_1'Y_2Y_2'Y_1) + (u_1 - u_2)'\left(2I_D - Y_1Y_1' - Y_2Y_2'\right)(u_1 - u_2)\right].$$

Ignoring the multiplicative factor, the first term is the same is the original Projection kernel, which we will denote as the 'linear' kernel to emphasize the underlying assumption:

$$k_{\mathrm{Lin}}(Y_1, Y_2) = \mathrm{tr}(Y_1Y_1'Y_2Y_2'), \tag{11}$$

The second term give rise to a new 'kernel'

$$k_u(Y_1, u_1, Y_2, u_2) = u_1'(2I_D - Y_1Y_1' - Y_2Y_2')u_2, \tag{12}$$

which measures the similarity of the offsets $u_1$ and $u_2$ scaled by $2I - Y_1Y_1' - Y_2Y_2'$. However, this term is not invariant under the invariance condition unfortunately. We instead propose the slight modification:

$$k(Y_1, u_1, Y_2, u_2) = u_1'(I - Y_1Y_1')(I - Y_2Y_2')u_2$$

The proof of the proposed form being invariant and positive definite is straightforward and is omitted. Combined with the linear term $k_{\mathrm{Lin}}$, this defines the new 'affine' kernel

$$k_{\mathrm{Aff}}(Y_1, u_1, Y_2, u_2) = \mathrm{tr}(Y_1Y_1'Y_2Y_2') + u_1'(I - Y_1Y_1')(I - Y_2Y_2')u_2. \tag{13}$$

As we can see, the KL distance with only the homogeneity condition has two terms related to the subspace $Y$ and the offset $u$. This suggests a general construction rule for affine kernels. If we have two separate positive kernels for subspaces and for offsets, we can add or multiply them together to construct new kernels [10].

## 3.2   Extension to scaled subspaces

We have assumed homogeneous subspace so far. However, if the subspaces are computed from the PCA of real data, the eigenvalues in general will have non-homogeneous values. To incorporate these scales for affine subspaces, we now allow the $Y$ to be non-orthonormal and check if the resultant kernel is still valid.

Let $Y_i$ be a full-rank $D \times m$ matrix, and $\widehat{Y}_i = Y_i(Y_i'Y_i)^{-1/2}$ be the orthonormalization of $Y_i$.

Ignoring the multiplicative factors, the limiting ($\sigma \rightarrow 0$) 'kernel' from (3) becomes

$$k = \frac{1}{2}\mathrm{tr}(\widehat{Y}_1\widehat{Y}_1'Y_2Y_2' + Y_1Y_1'\widehat{Y}_2\widehat{Y}_2') + u_1'(2I - \widehat{Y}_1\widehat{Y}_1' - \widehat{Y}_2\widehat{Y}_2')u_2,$$

which is again not well-defined.

The second term is the same as (12) in the previous subsection, and can be modified in the same way to $k_u = u_1'(I - \widehat{Y}_1\widehat{Y}_1')(I - \widehat{Y}_2\widehat{Y}_2')u_2$.

The first term is not positive definite, and there are several ways to remedy it. We propose to use the following form

$$k(Y_1, Y_2) = \frac{1}{2}\mathrm{tr}(Y_1\widehat{Y}_1'\widehat{Y}_2Y_2' + \widehat{Y}_1Y_1'Y_2\widehat{Y}_2') = \mathrm{tr}(\widehat{Y}_1'\widehat{Y}_2Y_2'Y_1),$$

among other possibilities.

The sum of the two modified terms, is the proposed 'affine scaled' kernel:

$$k_{\mathrm{AffSc}}(Y_1, u_1, Y_2, u_2) = \mathrm{tr}(Y_1\widehat{Y}_1'\widehat{Y}_2Y_2') + u_1'(I - \widehat{Y}_1\widehat{Y}_1')(I - \widehat{Y}_2\widehat{Y}_2')u_2. \tag{14}$$

This is a positive definite kernel which can be shown from the definition.

**Summary of the extended Projection kernels**

The proposed kernels are summarized below. Let $Y_i$ be a full-rank $D \times m$ matrix, and let $\widehat{Y}_i = Y_i(Y_i'Y_i)^{-1/2}$ the orthonormalization of $Y_i$ as before.

$$
\begin{aligned}
k_{\text{Lin}}(Y_1, Y_2) &= \text{tr}(\widehat{Y}_1'\widehat{Y}_2\widehat{Y}_2'\widehat{Y}_1), \quad k_{\text{LinSc}}(Y_1, Y_2) = \text{tr}(\widehat{Y}_1'\widehat{Y}_2 Y_2'Y_1) \\
k_{\text{Aff}}(Y_1, Y_2) &= \text{tr}(\widehat{Y}_1'\widehat{Y}_2\widehat{Y}_2'\widehat{Y}_1) + u_1'(I - \widehat{Y}_1\widehat{Y}_1')(I - \widehat{Y}_2\widehat{Y}_2')u_2 \\
k_{\text{AffSc}}(Y_1, Y_2) &= \text{tr}(\widehat{Y}_1'\widehat{Y}_2 Y_2'Y_1) + u_1'(I - \widehat{Y}_1\widehat{Y}_1')(I - \widehat{Y}_2\widehat{Y}_2')u_2
\end{aligned} \tag{15}
$$

We also spherize the kernels

$$
\widetilde{k}(Y_1, u_1, Y_2, u_2) = k(Y_1, u_1, Y_2, u_2)\, k(Y_1, u_1, Y_1, u_1)^{-1/2}\, k(Y_2, u_2, Y_2, u_2)^{-1/2}
$$

so that $k(Y_1, u_1, Y_1, u_1) = 1$ for any $Y_1$ and $u_1$.

There is a caveat in implementing these kernels. Although we used the same notations $Y$ and $\widehat{Y}$ for both linear and affine kernels, they are different in computation. For linear kernels the $Y$ and $\widehat{Y}$ are computed from data assuming $u = 0$, whereas for affine kernels the $Y$ and $\widehat{Y}$ are computed after removing the estimated mean $u$ from the data.

## 3.3 Extension to nonlinear subspaces

A systematic way of extending the Projection kernel from linear/affine subspaces to nonlinear spaces is to use an implicit map via a kernel function, where the latter kernel is to be distinguished from the former kernels. Note that the proposed kernels (15) can be computed only from the inner products of the column vectors of $Y$'s and $u$'s including the orthonormalization procedure. If we replace the inner products of those vectors $y_i'y_i$ by a positive definite function $f(y_i, y_j)$ on Euclidean spaces, this implicitly defines a nonlinear feature space. This 'doubly kernel' approach has already been proposed for the Binet-Cauchy kernel [17, 8] and for probabilistic distances in general [18]. We can adopt the trick for the extended Projection kernels as well to extend the kernels to operate on 'nonlinear subspaces'[3].

# 4 Experiments with synthetic data

In this section we demonstrate the application of the extended Projection kernels to two-class classification problems with Support Vector Machines (SVMs).

## 4.1 Synthetic data

The extended kernels are defined under different assumptions of data distribution. To test the kernels we generate three types of data – 'easy', 'intermediate' and 'difficult' – from MFA distribution, which cover the different ranges of data distribution.

A total of $N = 100$ FA distributions are generated in $D = 10$ dimensional space. The parameters of each FA distribution $p_i(x) = \mathcal{N}(u_i, C_i)$ are randomly chosen such that

- 'Easy' data have well separarted means $u_i$ and homogeneous scales $Y_i'Y_i$
- 'Intermediate' data have partially overlapping means $u_i$ and homogeneous scales $Y_i'Y_i$
- 'Difficult' data have totally overlapping means ($u_1 = ... = u_N = 0$) and randomly chosen scales between 0 and 1.

The class label for each distribution $p_i$ is assigned as follows. We choose a pair of distribution $p_+$ and $p_-$ which are the farthest apart from each other among all pairs of distributions. Then the labels of the remaining distributions $p_i$ are determined from whether they are close to $p_+$ or $p_-$. The distances are measured by the KL distance $J_{\text{KL}}$.

## 4.2 Algorithms and results

We compare the performance of the Euclidean SVM with linear/ polynomial/ RBF kernels and the performance of SVM with Grassmann kernels. To test the original SVMs, we randomly sampled $n = 50$ point from each FA distribution $p_i(x)$. We evaluate the algorithm with $N$-fold cross validation by holding out one set and training with the other $N - 1$ sets. The polynomial kernel used is $k(x_1, x_2) = (\langle x_1, x_2 \rangle + 1)^3$.

To test the Grassmann SVM, we first estimated the mean $u_i$ and the basis $Y_i$ from $n = 50$ points of each FA distribution $p_i(x)$ used for the original SVM. The $Y_i$, $\mu_i$ and $\sigma$ are estimated simply from the probabilistic PCA [12], although they can also be estimated by the Expectation Maximization approach.

Six different Grassmann kernels are compared: 1) the original and the extended Projection kernels (Linear, Linear Scaled, Affine, Affine Scaled), 2) the Binet-Cauchy kernel $k_{\text{BC}}(Y_1, Y_2) = (\det Y_1' Y_2)^2 = \det Y_1' Y_2 Y_2' Y_1$, and 3) the Bhattacharyya kernel $k_{\text{Bhat}}(p_1, p_2) = \int [p_1(x) \, p_2(x)]^{1/2} \, dx$ adapted for FA distributions. We evaluate the algorithms with leave-one-out test by holding out one subspace and training with the other $N - 1$ subspaces.

Table 1: Classification rates of the Euclidean SVMs and the Grassmann SVMs. The BC and Bhat are short for Binet-Cauchy and Bhattacharyya kernels, respectively.

| | Euclidean | | Grassmann | | | | | Probabilistic |
|---|---|---|---|---|---|---|---|---|
| | Linear | Poly | Linear | Lin Sc | Aff | Aff Sc | BC | Bhat |
| *Easy* | 84.63 | 79.85 | 55.10 | 55.30 | 92.70 | 92.30 | 54.70 | 46.10 |
| *Intermediate* | 62.40 | 61.76 | 68.10 | 67.50 | 85.20 | 83.60 | 60.90 | 59.00 |
| *Difficult* | 52.00 | 63.74 | 80.10 | 81.00 | 80.30 | 81.20 | 68.90 | 77.30 |

Table 1 shows the classification rates of the Euclidean SVMs and the Grassmann SVMs, averaged for 10 trials. The results shows that best rates are obtained from the extended kernels, and the Euclidean kernels lag behind for all three types of data. Interestingly the polynomial kernels often perform worse than the linear kernels, and the RBF kernel performed even worse which we do not report. For the 'difficult' data where the means are zero, the linear SVMs degrade to the chance-level (50%), which agrees with the intuitive picture that any decision hyperplane that passes the origin will roughly halve the points from a zero-mean distribution. As expected, the linear kernel is inappropriate for data with nonzero offsets ('easy' and 'intermediate'), whereas the affine kernel performs well regardless of the offsets. However, there is no significant difference between the homogeneous and the scaled kernels. The Binet-Cauchy and the Bhattacharyya kernels mostly underperformed.

We conclude that under certain conditions the extended kernels have clear advantages over the original linear kernels and the Euclidean kernels for the subspace-based classification problem.

## 5 Experiments with real-world data

In this section we demonstrate the application of the extended Projection kernels to recognition problems with the kernel Fisher Discriminant Analysis [10].

### 5.1 Databases

The Yale face database and the Extended Yale face database [3] together consist of pictures of 38 subjects with 9 different poses and 45 different lighting conditions. The ETH-80 [9] is an object database designed for object categorization test under varying poses. The database consists of pictures of 8 object categories and 10 object instances for each category, recored under 41 different poses.

These databases have naturally factorized structures which make them ideal to test subspace-based learning algorithms with. In Yale Face database, a set consists of images of all illumination conditions a person at a fixed pose. By treating the set as a point in the Grassmann manifold, we can perform illumination-invarint learning tasks with the data. For ETH-80 database, a set consists of images of all possible poses of an object from a category. Also by treating such set as a point in the Grassmann manifold, we can perform pose-invariant learning tasks with the data.

There are a total of $N = 279$ and 80 sets as described above respectively. The images are resized to the dimension of $D = 504$ and 1024 respectively, and the maximum of $m = 9$ dimensional subspaces are used to compute the kernels. The subspace parameters $Y_i$, $u_i$ and $\sigma$ are estimated from the probabilistic PCA [12].

## 5.2 Algorithms and results

We perform subject recognition tests with Yale Face, and categorization tests with ETH-80 database. Since these databases are highly multiclass (31 and 8 classes) relative to the total number of samples, we use the kernel Discriminant Analysis to reduce dimensionality and extract features, in conjunction with a 1-NN classifier. The six different Grassmann kernels are compared: the extended Projection (Lin/LinSc/Aff/Affsc) kernels, the Binet-Cauchy kernel, and the Bhattacharyya kernel. The baseline algorithm (Eucl) is the Linear Discriminant Analysis applied to the original images in the data from which the subspaces are computed.

Figure 1 summarizes the average recognition/categoriazation rates from 9- and 10-fold cross validation with the Yale Face and ETH-80 databases respectively. The results shows that best rates are achieved from the extended kernels: linear scaled kernel in Yale Face and the affine kernel in ETH-80. However the difference within the extended kernels are small. The performance of the extended kernels remain relatively unaffected by the subspace dimensionality, which is a convenient property in practice since we do not know the true dimensionality a priori. However the Binet-Cauchy and the Bhattacharyya kernels do not perform as well, and degrade fast as the subspace dimension increases. The analysis of the poor performance are given in the thesis [5].

## 6 Conclusion

In this paper we analyzed the relationship between probabilistic distances and the geometric Grassmann kernels, especially the KL distance and the Projection kernel. This analysis help us to understand the limitations of the Bhattacharyya kernel for subspace-based problems, and also suggest the extensions of the Projection kernel. With synthetic and real data we demonstrated that the extended kernels can outperform the original Projection kernel, as well as the previously used Bhattacharyya and the Binet-Cauchy kernels for subspace-based classification problems. The relationship between other probabilistic distances and the Grassmann kernels is yet to be fully explored, and we expect to see more results from a follow-up study.

## Footnotes

[1]by distance we mean any nonnegative measure of similarity and not necessarily a metric.

[2]The Grassmann manifold is defined as a quotient space $\mathbb{O}(D)/\mathbb{O}(m) \times \mathbb{O}(D-m)$ where $\mathbb{O}$ is the orthogonal group. The affine Grassmann manifold is similarly defined as $\mathbb{E}(D)/\mathbb{E}(m) \times \mathbb{O}(D-m)$, where $\mathbb{E}$ is the Euclidean group. Fore more explanations, please refer to [5].

[3]the preimage corresponding to the linear subspaces in the RKHS via the feature map

## References

[1] Gianfranco Doretto, Alessandro Chiuso, Ying Nian Wu, and Stefano Soatto. Dynamic textures. *Int. J. Comput. Vision*, 51(2):91–109, 2003.

[2] Alan Edelman, Tomás A. Arias, and Steven T. Smith. The geometry of algorithms with orthogonality constraints. *SIAM J. Matrix Anal. Appl.*, 20(2):303–353, 1999.

[3] Athinodoros S. Georghiades, Peter N. Belhumeur, and David J. Kriegman. From few to many: Illumination cone models for face recognition under variable lighting and pose. *IEEE Trans. Pattern Anal. Mach. Intell.*, 23(6):643–660, 2001.

[4] Zoubin Ghahramani and Geoffrey E. Hinton. The EM algorithm for mixtures of factor analyzers. Technical Report CRG-TR-96-1, Department of Computer Science, University of Toronto, 21 1996.

[5] Jihun Hamm. *Subspace-based Learning with Grassmann Manifolds*. Ph.D thesis in Electrical and Systems Engineering, University of Pennsylvania, 2008. Available at http://www.seas.upenn.edu/ jhham/Papers/thesis-jh.pdf.

[6] Jihun Hamm and Daniel Lee. Grassmann discriminant analysis: a unifying view on subspace-based learning. In *Int. Conf. Mach. Learning*, 2008.

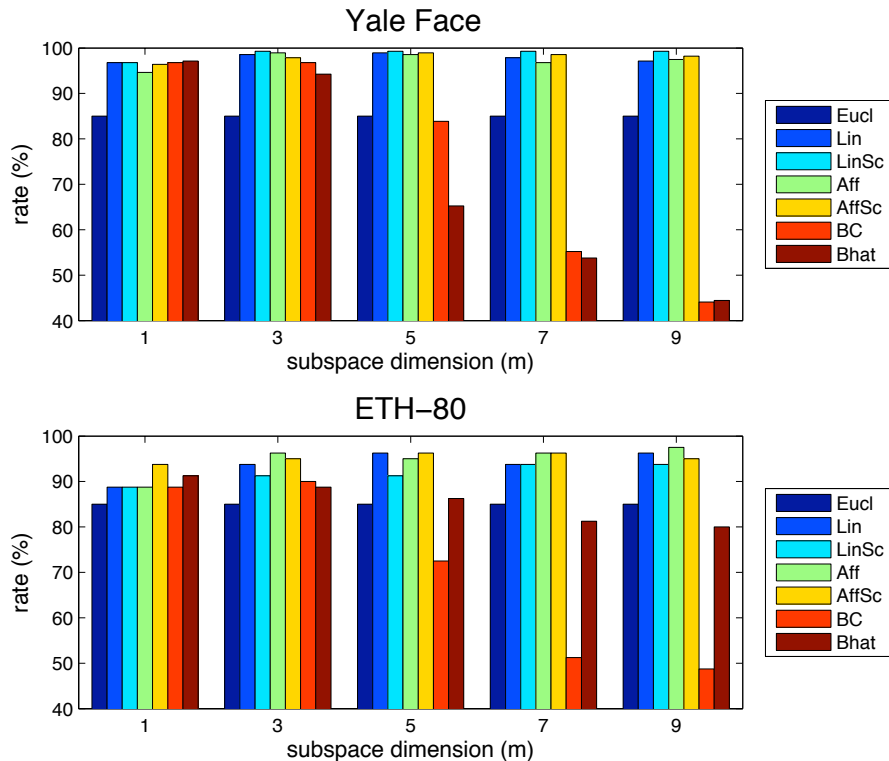

Figure 1: Comparison of Grassmann kernels for face recognition/ object categorization tasks with kernel discriminant analysis. The extended Projection kernels (Lin/LinSc/Aff/ AffSc) outperform the baseline method (Eucl) and the Binet-Cauchy (BC) and the Bhattacharyya (Bhat) kernels.

[7]  Tony Jebara and Risi Imre Kondor. Bhattacharyya expected likelihood kernels. In *COLT*, pages 57–71, 2003.

[8]  Risi Imre Kondor and Tony Jebara. A kernel between sets of vectors. In *Proc. of the 20th Int. Conf. on Mach. Learn.*, pages 361–368, 2003.

[9]  Bastian Leibe and Bernt Schiele. Analyzing appearance and contour based methods for object categorization. *CVPR*, 02:409, 2003.

[10]  Bernhard Schölkopf and Alexander J. Smola. *Learning with Kernels: Support Vector Machines, Regularization, Optimization, and Beyond*. MIT Press, Cambridge, MA, USA, 2001.

[11]  Gregory Shakhnarovich, John W. Fisher, and Trevor Darrell. Face recognition from long-term observations. In *Proc. of the 7th Euro. Conf. on Computer Vision*, pages 851–868, London, UK, 2002.

[12]  Michael E. Tipping and Christopher M. Bishop. Probabilistic principal component analysis. *Journal Of The Royal Statistical Society Series B*, 61(3):611–622, 1999.

[13]  Pavan Turaga, Ashok Veeraraghavan, and Rama Chellappa. Statistical analysis on Stiefel and Grassmann manifolds with applications in computer vision. In *CVPR*, 2008.

[14]  Ashok Veeraraghavan, Amit K. Roy-Chowdhury, and Rama Chellappa. Matching shape sequences in video with applications in human movement analysis. *IEEE Trans. Pattern Anal. Mach. Intell.*, 27(12):1896–1909, 2005.

[15]  S.V.N. Vishwanathan and Alexander J. Smola. Binet-Cauchy kernels. In *NIPS*, 2004.

[16]  Liwei Wang, Xiao Wang, and Jufu Feng. Subspace distance analysis with application to adaptive bayesian algorithm for face recognition. *Pattern Recogn.*, 39(3):456–464, 2006.

[17]  Lior Wolf and Amnon Shashua. Learning over sets using kernel principal angles. *J. Mach. Learn. Res.*, 4:913–931, 2003.

[18]  Shaohua Kevin Zhou and Rama Chellappa. From sample similarity to ensemble similarity: Probabilistic distance measures in Reproducing Kernel Hilbert Space. *IEEE Trans. Pattern Anal. Mach. Intell.*, 28(6):917–929, 2006.

